# The Doubly Correlated Nonparametric Topic Model

**Dae Il Kim and Erik B. Sudderth**
Department of Computer Science
Brown University, Providence, RI 02906
daeil@cs.brown.edu, sudderth@cs.brown.edu

## Abstract

Topic models are learned via a statistical model of variation within document collections, but designed to extract meaningful semantic structure. Desirable traits include the ability to incorporate annotations or metadata associated with documents; the discovery of correlated patterns of topic usage; and the avoidance of parametric assumptions, such as manual specification of the number of topics. We propose a *doubly correlated nonparametric topic* (DCNT) model, the first model to simultaneously capture all three of these properties. The DCNT models metadata via a flexible, Gaussian regression on arbitrary input features; correlations via a scalable square-root covariance representation; and nonparametric selection from an unbounded series of potential topics via a stick-breaking construction. We validate the semantic structure and predictive performance of the DCNT using a corpus of NIPS documents annotated by various metadata.

## 1 Introduction

The contemporary problem of exploring huge collections of discrete data, from biological sequences to text documents, has prompted the development of increasingly sophisticated statistical models. Probabilistic topic models represent documents via a mixture of topics, which are themselves distributions on the discrete vocabulary of the corpus. *Latent Dirichlet allocation* (LDA) [3] was the first hierarchical Bayesian topic model, and remains influential and widely used. However, it suffers from three key limitations which are jointly addressed by our proposed model.

The first assumption springs from LDA's Dirichlet prior, which implicitly neglects correlations[1] in document-specific topic usage. In diverse corpora, true semantic topics may exhibit strong (positive or negative) correlations; neglecting these dependencies may distort the inferred topic structure. The *correlated topic model* (CTM) [2] uses a logistic-normal prior to express correlations via a latent Gaussian distribution. However, its usage of a "soft-max" (multinomial logistic) transformation requires a global normalization, which in turn presumes a fixed, finite number of topics.

The second assumption is that each document is represented solely by an unordered "bag of words". However, text data is often accompanied by a rich set of metadata such as author names, publication dates, relevant keywords, etc. Topics that are consistent with such metadata may also be more semantically relevant. The *Dirichlet multinomial regression* (DMR) [11] model conditions LDA's Dirichlet parameters on feature-dependent linear regressions; this allows metadata-specific topic frequencies but retains other limitations of the Dirichlet. Recently, the *Gaussian process topic model* [1] incorporated correlations at the topic level via a topic covariance, and the document level via an appropriate GP kernel function. This model remains parametric in its treatment of the number of topics, and computational scaling to large datasets is challenging since learning scales super-linearly with the number of documents.

The third assumption is the *a priori* choice of the number of topics. The most direct nonparametric extension of LDA is the *hierarchical Dirichlet process* (HDP) [17]. The HDP allows an unbounded set of topics via a latent stochastic process, but nevertheless imposes a Dirichlet distribution on any finite subset of these topics. Alternatively, the *nonparametric Bayes pachinko allocation* [9] model captures correlations within an unbounded topic collection via an inferred, directed acyclic graph. More recently, the *discrete infinite logistic normal* [13] (DILN) model of topic correlations used an exponentiated Gaussian process (GP) to rescale the HDP. This construction is based on the gamma process representation of the DP [5]. While our goals are similar, we propose a rather different model based on the stick-breaking representation of the DP [16]. This choice leads to arguably simpler learning algorithms, and also facilitates our modeling of document metadata.

In this paper, we develop a *doubly correlated nonparametric topic* (DCNT) model which captures between-topic correlations, as well as between-document correlations induced by metadata, for an unbounded set of potential topics. As described in Sec. 2, the global soft-max transformation of the DMR and CTM is replaced by a stick-breaking transformation, with inputs determined via both metadata-dependent linear regressions and a square-root covariance representation. Together, these choices lead to a well-posed nonparametric model which allows tractable MCMC learning and inference (Sec. 3). In Sec. 4, we validate the model using a toy dataset, as well as a corpus of NIPS documents annotated by author and year of publication.

## 2 A Doubly Correlated Nonparametric Topic Model

The DCNT is a hierarchical, Bayesian nonparametric generalization of LDA. Here we give an overview of the model structure (see Fig. 1), focusing on our three key innovations.

### 2.1 Document Metadata

Consider a collection of $D$ documents. Let $\phi_d \in \mathbb{R}^F$ denote a feature vector capturing the metadata associated with document $d$, and $\phi$ an $F \times D$ matrix of corpus metadata. When metadata is unavailable, we assume $\phi_d = 1$. For each of an unbounded sequence of topics $k$, let $\eta_{fk} \in \mathbb{R}$ denote an associated significance weight for feature $f$, and $\eta_{:k} \in \mathbb{R}^F$ a vector of these weights.[2]

We place a Gaussian prior $\eta_{:k} \sim N(\mu, \Lambda^{-1})$ on each topic's weights, where $\mu \in \mathbb{R}^F$ is a vector of mean feature responses, and $\Lambda$ is an $F \times F$ diagonal precision matrix. In a hierarchical Bayesian fashion [6], these parameters have priors $\mu_f \sim N(0, \gamma_\mu)$, $\lambda_f \sim \text{Gam}(a_f, b_f)$. Appropriate values for the hyperparameters $\gamma_\mu$, $a_f$, and $b_f$ are discussed later.

Given $\eta$ and $\phi_d$, the document-specific "score" for topic $k$ is sampled as $u_{kd} \sim N(\eta_{:k}^T \phi_d, 1)$. These real-valued scores are mapped to document-specific topic frequencies $\pi_{kd}$ in subsequent sections.

### 2.2 Topic Correlations

For topic $k$ in the ordered sequence of topics, we define a sequence of $k$ linear transformation weights $A_{k\ell}$, $\ell = 1, \ldots, k$. We then sample a variable $v_{kd}$ as follows:

$$v_{kd} \sim N\left( \sum_{\ell=1}^{k} A_{k\ell} u_{\ell d}, \lambda_v^{-1} \right) \tag{1}$$

Let $A$ denote a lower triangular matrix containing these values $A_{k\ell}$, padded by zeros. Slightly abusing notation, we can then compactly write this transformation as $v_{:d} \sim N(A u_{:d}, L^{-1})$, where $L = \lambda_v I$ is an infinite diagonal precision matrix. Critically, note that the distribution of $v_{kd}$ depends only on the first $k$ entries of $u_{:d}$, not the infinite tail of scores for subsequent topics.

Marginalizing $u_{:d}$, the covariance of $v_{:d}$ equals $\text{Cov}[v_{:d}] = AA^T + L^{-1} \triangleq \Sigma$. As in the classical factor analysis model, $A$ encodes a square-root representation of an output covariance matrix. Our integration of input metadata has close connections to the semiparametric latent factor model [18], but we replace their kernel-based GP covariance representation with a feature-based regression.

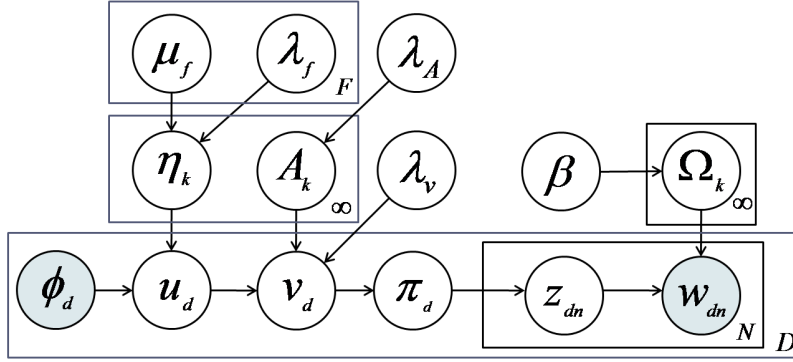

Figure 1: Directed graphical representation of a DCNT model for $D$ documents containing $N$ words. Each of the unbounded set of topics has a word distribution $\Omega_k$. The topic assignment $z_{dn}$ for word $w_{dn}$ depends on document-specific topic frequencies $\pi_d$, which have a correlated dependence on the metadata $\phi_d$ produced by $A$ and $\eta$. The Gaussian latent variables $u_d$ and $v_d$ implement this mapping, and simplify MCMC methods.

Given similar lower triangular representations of factorized covariance matrices, conventional Bayesian factor analysis models place a symmetric Gaussian prior $A_{k\ell} \sim N(0, \lambda_A^{-1})$. Under this prior, however, $\mathbb{E}[\Sigma_{kk}] = k\lambda_A^{-1}$ grows linearly with $k$. This can produce artifacts for standard factor analysis [10], and is disastrous for the DCNT where $k$ is unbounded. We instead propose an alternative prior $A_{k\ell} \sim N(0, (k\lambda_A)^{-1})$, so that the variance of entries in the $k^{th}$ row is reduced by a factor of $k$. This shrinkage is carefully chosen so that $\mathbb{E}[\Sigma_{kk}] = \lambda_A^{-1}$ remains constant.

If we constrain $A$ to be a diagonal matrix, with $A_{kk} \sim N(0, \lambda_A^{-1})$ and $A_{k\ell} = 0$ for $k \neq \ell$, we recover a simplified *singly correlated nonparametric topic* (SCNT) model which captures metadata but not topic correlations. For either model, the precision parameters are assigned conjugate gamma priors $\lambda_v \sim \mathrm{Gam}(a_v, b_v)$, $\lambda_A \sim \mathrm{Gam}(a_A, b_A)$.

### 2.3 Logistic Mapping to Stick-Breaking Topic Frequencies

Stick breaking representations are widely used in applications of nonparametric Bayesian models, and lead to convenient sampling algorithms [8]. Let $\pi_{kd}$ be the probability of choosing topic $k$ in document $d$, where $\sum_{k=1}^{\infty} \pi_{kd} = 1$. The DCNT constructs these probabilities as follows:

$$\pi_{kd} = \psi(v_{kd}) \prod_{\ell=1}^{k-1} \psi(-v_{\ell d}), \qquad \psi(v_{kd}) = \frac{1}{1 + \exp(-v_{kd})}. \tag{2}$$

Here, $0 < \psi(v_{kd}) < 1$ is the classic logistic function, which satisfies $\psi(-v_{\ell d}) = 1 - \psi(v_{\ell d})$. This same transformation is part of the so-called *logistic stick-breaking process* [14], but that model is motivated by different applications, and thus employs a very different prior distribution for $v_{kd}$.

Given the distribution $\pi_{:d}$, the topic assignment indicator for word $n$ in document $d$ is drawn according to $z_{dn} \sim \mathrm{Mult}(\pi_{:d})$. Finally, $w_{dn} \sim \mathrm{Mult}(\Omega_{z_{dn}})$ where $\Omega_k \sim \mathrm{Dir}(\beta)$ is the word distribution for topic $k$, sampled from a Dirichlet prior with symmetric hyperparameters $\beta$.

## 3 Monte Carlo Learning and Inference

We use a Markov chain Monte Carlo (MCMC) method to approximately sample from the posterior distribution of the DCNT. For most parameters, our choice of conditionally conjugate priors leads to closed form Gibbs sampling updates. Due to the logistic stick-breaking transformation, closed form resampling of $v$ is intractable; we instead use a Metropolis independence sampler [6].

Our sampler is based on a finite *truncation* of the full DCNT model, which has proven useful with other stick-breaking priors [8, 14, 15]. Let $K$ be the maximum number topics. As our experiments demonstrate, $K$ is not the number of topics that will be utilized by the learned model, but rather a (possibly loose) upper bound on that number. For notational convenience, let $\bar{K} = K - 1$.

Under the truncated model, $\eta$ is a $F \times \bar{K}$ matrix of regression coefficients, and $u$ is a $\bar{K} \times D$ matrix satisfying $u_{:d} \sim N(\eta^T \phi_d, I_{\bar{K}})$. Similarly, $A$ is a $\bar{K} \times \bar{K}$ lower triangular matrix, and $v_{:d} \sim N(Au_{:d}, \lambda_v^{-1} I_{\bar{K}})$. The probabilities $\pi_{kd}$ for the first $\bar{K}$ topics are set as in eq. (2), with the final topic set so that a valid distribution is ensured: $\pi_{Kd} = 1 - \sum_{k=1}^{K-1} \pi_{kd} = \prod_{k=1}^{K-1} \psi(-v_{kd})$.

## 3.1 Gibbs Updates for Topic Assignments, Correlation Parameters, and Hyperparameters

The precision parameter $\lambda_f$ controls the variability of the feature weights associated with each topic. As in many regression models, the gamma prior is conjugate so that

$$p(\lambda_f \mid \eta, a_f, b_f) \propto \mathrm{Gam}(\lambda_f \mid a_f, b_f) \prod_{k=1}^{\bar{K}} N(\eta_{fk} \mid \mu_f, \lambda_f^{-1})$$

$$\propto \mathrm{Gam}\left(\lambda_f \mid \frac{1}{2}\bar{K} + a_f, \ \frac{1}{2}\sum_{k=1}^{\bar{K}}(\eta_{fk} - \mu_f)^2 + b_f\right). \tag{3}$$

Similarly, the precision parameter $\lambda_v$ has a gamma prior and posterior:

$$p(\lambda_v \mid v, a_v, b_v) \propto \mathrm{Gam}(\lambda_v \mid a_v, b_v) \prod_{d=1}^{D} N(v_{:d} \mid Au_{:d}, L^{-1})$$

$$\propto \mathrm{Gam}\left(\lambda_v \mid \frac{1}{2}\bar{K}D + a_v, \ \frac{1}{2}\sum_{d=1}^{D}(v_{:d} - Au_{:d})^T(v_{:d} - Au_{:d}) + b_v\right). \tag{4}$$

Entries of the regression matrix $A$ have a rescaled Gaussian prior $A_{k\ell} \sim N(0, (k\lambda_A)^{-1})$. With a gamma prior, the precision parameter $\lambda_A$ nevertheless has the following gamma posterior:

$$p(\lambda_A \mid A, a_A, b_A) \propto \mathrm{Gam}(\lambda_A \mid a_A, b_A) \prod_{k=1}^{\bar{K}} \prod_{\ell=1}^{k} N(A_{k\ell} \mid 0, (k\lambda_A)^{-1})$$

$$\propto \mathrm{Gam}\left(\lambda_A \mid \frac{1}{2}\bar{K}(\bar{K} - 1) + a_A, \ \frac{1}{2}\sum_{k=1}^{\bar{K}}\sum_{\ell=1}^{k} k A_{k\ell}^2 + b_A\right). \tag{5}$$

Conditioning on the feature regression weights $\eta$, the mean weight $\mu_f$ in our hierarchical prior for each feature $f$ has a Gaussian posterior:

$$p(\mu_f \mid \eta) \propto N(\mu_f \mid 0, \gamma_\mu) \prod_{k=1}^{\bar{K}} N(\eta_{fk} \mid \mu_f, \lambda_f^{-1})$$

$$\propto N\left(\mu_f \mid \frac{\gamma_\mu}{\bar{K}\gamma_\mu + \lambda_f^{-1}} \sum_{k=1}^{\bar{K}} \eta_{fk}, \ (\gamma_\mu^{-1} + \bar{K}\lambda_f)^{-1}\right) \tag{6}$$

To sample $\eta_{:k}$, the linear function relating metadata to topic $k$, we condition on all documents $u_{k:}$ as well as $\phi$, $\mu$, and $\Lambda$. Columns of $\eta$ are conditionally independent, with Gaussian posteriors:

$$p(\eta_{:k} \mid u, \phi, \mu, \Lambda) \propto N(\eta_{:k} \mid \mu, \Lambda^{-1}) N(u_{k:}^T \mid \phi^T \eta_{:k}, I_D)$$
$$\propto N(\eta_{:k} \mid (\Lambda + \phi\phi^T)^{-1}(\phi u_{k:}^T + \Lambda\mu), \ (\Lambda + \phi\phi^T)^{-1}). \tag{7}$$

Similarly, the scores $u_{:d}$ for each document are conditionally independent with Gaussian posteriors:

$$p(u_{:d} \mid v_{:d}, \eta, \phi_d, L) \propto N(u_{:d} \mid \eta^T \phi_d, I_{\bar{K}}) N(v_{:d} \mid Au_{:d}, L^{-1})$$
$$\propto N(u_{:d} \mid (I_{\bar{K}} + A^T LA)^{-1}(A^T L v_{:d} + \eta^T \phi_d), \ (I_{\bar{K}} + A^T LA)^{-1}). \tag{8}$$

To resample $A$, we note that its rows are conditionally independent. The posterior of the $k$ entries $A_{k:}$ in row $k$ depends on $v_{k:}$ and $\hat{U}_k \triangleq u_{1:k,:}$, the first $k$ entries of $u_{:d}$ for each document $d$:

$$p(A_{k:}^T \mid v_{k:}, \hat{U}_k, \lambda_A, \lambda_v) \propto \prod_{j=1}^{k} N(A_{kj} \mid 0, (k\lambda_A)^{-1}) N(v_{k:}^T \mid \hat{U}_k^T A_{k:}^T, \lambda_v^{-1} I_D) \tag{9}$$

$$\propto N(A_{k:}^T \mid (k\lambda_A \lambda_v^{-1} I_k + \hat{U}_k \hat{U}_k^T)^{-1} \hat{U}_k v_{k:}^T, \ (k\lambda_A I_k + \lambda_v \hat{U}_k \hat{U}_k^T)^{-1}).$$

For the SCNT model, there is a related but simpler update (see supplemental material).

As in collapsed sampling algorithms for LDA [7], we can analytically marginalize the word distribution $\Omega_k$ for each topic. Let $M_{kw}^{\backslash dn}$ denote the number of instances of word $w$ assigned to topic $k$, excluding token $n$ in document $d$, and $M_{k.}^{\backslash dn}$ the number of total tokens assigned to topic $k$. For a vocabulary with $W$ unique word types, the posterior distribution of topic indicator $z_{dn}$ is then

$$p(z_{dn} = k \mid \pi_{:d}, z_{\backslash dn}) \propto \pi_{kd} \left( \frac{M_{kw}^{\backslash dn} + \beta}{M_{k.}^{\backslash dn} + W\beta} \right). \tag{10}$$

Recall that the topic probabilities $\pi_{:d}$ are determined from $v_{:d}$ via Equation (2).

### 3.2 Metropolis Independence Sampler Updates for Topic Activations

The posterior distribution of $v_{:d}$ does not have a closed analytical form due to the logistic nonlinearity underlying our stick-breaking construction. We instead employ a Metropolis-Hastings independence sampler, where proposals $q(v_{:d}^* \mid v_{:d}, A, u_{:d}, \lambda_v) = N(v_{:d}^* \mid Au_{:d}, \lambda_v^{-1} I_{\bar{K}})$ are drawn from the prior. Combining this with the likelihood of the $N_d$ word tokens, the proposal is accepted with probability $\min(\mathbb{A}(v_{:d}^*, v_{:d}), 1)$, where

$$
\begin{aligned}
\mathbb{A}(v_{:d}^*, v_{:d}) &= \frac{p(v_{:d}^* \mid A, u_{:d}, \lambda_v) \prod_{n=1}^{N_d} p(z_{dn} \mid v_{:d}^*) q(v_{:d} \mid v_{:d}^*, A, u_{:d}, \lambda_v)}{p(v_{:d} \mid A, u_{:d}, \lambda_v) \prod_{n=1}^{N_d} p(z_{dn} \mid v_{:d}) q(v_{:d}^* \mid v_{:d}, A, u_{:d}, \lambda_v)} \\
&= \prod_{n=1}^{N_d} \frac{p(z_{dn} \mid v_{:d}^*)}{p(z_{dn} \mid v_{:d})} = \prod_{k=1}^{K} \left( \frac{\pi_{kd}^*}{\pi_{kd}} \right)^{\sum_{n=1}^{N_d} \delta(z_{dn}, k)}
\end{aligned}
\tag{11}
$$

Because the proposal cancels with the prior distribution in the acceptance ratio $\mathbb{A}(v_{:d}^*, v_{:d})$, the final probability depends only on a ratio of likelihood functions, which can be easily evaluated from counts of the number of words assigned to each topic by $z_d$.

## 4 Experimental Results

### 4.1 Toy Bars Dataset

Following related validations of the LDA model [7], we ran experiments on a toy corpus of "images" designed to validate the features of the DCNT. The dataset consisted of 1,500 images (documents), each containing a vocabulary of 25 pixels (word types) arranged in a 5x5 grid. Documents can be visualized by displaying pixels with intensity proportional to the number of corresponding words (see Figure 2). Each training document contained 300 word tokens.

Ten topics were defined, corresponding to all possible horizontal and vertical 5-pixel "bars". We consider two toy datasets. In the first, a random number of topics is chosen for each document, and then a corresponding subset of the bars is picked uniformly at random. In the second, we induce topic correlations by generating documents that contain a combination of either only horizontal (topics 1-5) or only vertical (topics 6-10) bars. For these datasets, there was no associated metadata, so the input features were simply set as $\phi_d = 1$.

Using these toy datasets, we compared the LDA model to several versions of the DCNT. For LDA, we set the number of topics to the true value of $K = 10$. Similar to previous toy experiments [7], we set the parameters of its Dirichlet prior over topic distributions to $\alpha = 50/K$, and the topic smoothing parameter to $\beta = 0.01$. For the DCNT model, we set $\gamma_\mu = 10^6$, and all gamma prior hyperparameters as $a = b = 0.01$, corresponding to a mean of 1 and a variance of 100. To initialize the sampler, we set the precision parameters to their prior mean of 1, and sample all other variables from their prior. We compared three variants of the DCNT model: the singly correlated SCNT ($A$ constrained to be diagonal) with $K = 10$, the DCNT with $K = 10$, and the DCNT with $K = 20$. The final case explores whether our stick-breaking prior can successfully infer the number of topics.

For the toy dataset with correlated topics, the results of running all sampling algorithms for 10,000 iterations are illustrated in Figure 2. On this relatively clean data, all models limited to $K = 10$

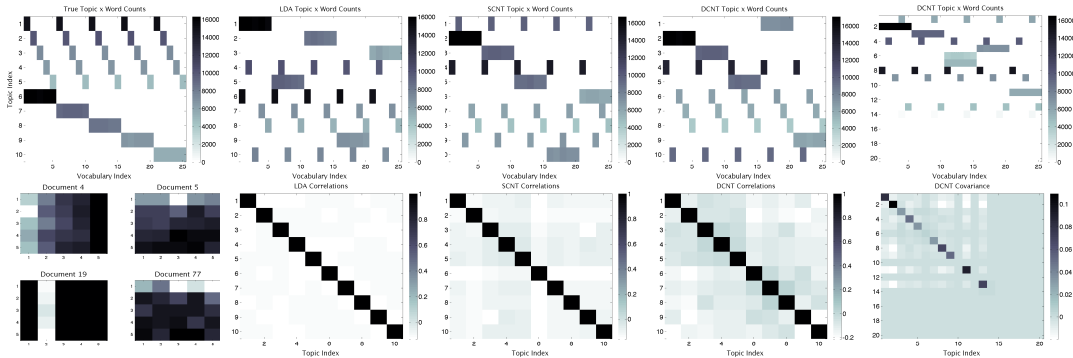

Figure 2: A dataset of correlated toy bars (example document images in bottom left). *Top:* From left to right, the true counts of words generated by each topic, and the recovered counts for LDA ($K = 10$), SCNT ($K = 10$), DCNT ($K = 10$), and DCNT ($K = 20$). Note that the true topic order is not identifiable. *Bottom:* Inferred topic covariance matrices for the four corresponding models. Note that LDA assumes all topics have a slight negative correlation, while the DCNT infers more pronounced positive correlations. With $K = 20$ potential DCNT topics, several are inferred to be unused with high probability, and thus have low variance.

topics recover the correct topics. With $K = 20$ topics, the DCNT recovers the true topics, as well as a redundant copy of one of the bars. This is typical behavior for sampling runs of this length; more extended runs usually merge such redundant bars. The development of more rapidly mixing MCMC methods is an interesting area for future research.

To determine the topic correlations corresponding to a set of learned model parameters, we use a Monte Carlo estimate (details in the supplemental material). To make these matrices easier to visualize, the Hungarian algorithm was used to reorder topic labels for best alignment with the ground truth topic assignments. Note the significant blocks of positive correlations recovered by the DCNT, reflecting the true correlations used to create this toy data.

## 4.2 NIPS Corpus

The NIPS corpus that we used consisted of publications from previous NIPS conferences 0-12 (1987-1999), including various metadata (year of publication, authors, and section categories). We compared four variants of the DCNT model: a model which ignored metadata, a model with indicator features for the year of publication, a model with indicator features for year of publication and the presence of highly prolific authors (those with more than 10 publications), and a model with features for year of publication and additional authors (those with more than 5 publications). In all cases, the feature matrix $\phi$ is binary. All models were truncated to use at most $K = 50$ topics, and the sampler initialized as in Sec. 4.1.

### 4.2.1 Conditioning on Metadata

A learned DCNT model provides predictions for how topic frequencies change given particular metadata associated with a document. In Figure 3, we show how predicted topic frequencies change over time, conditioning also on one of three authors (Michael Jordan, Geoffrey Hinton, or Terrence Sejnowski). For each, words from a relevant topic illustrate how conditioning on a particular author can change the predicted document content. For example, the visualization associated with Michael Jordan shows that the frequency of the topic associated with probabilistic models gradually increases over the years, while the topic associated with neural networks decreases. Conditioning on Geoffrey Hinton puts larger mass on a topic which focuses on models developed by his research group. Finally, conditioning on Terrence Sejnowski dramatically increases the probability of topics related to neuroscience.

### 4.2.2 Correlations between Topics

The DCNT model can also capture correlations between topics. In Fig. 4, we visualize this using a diagram where the size of a colored grid is proportional to the magnitude of the correlation

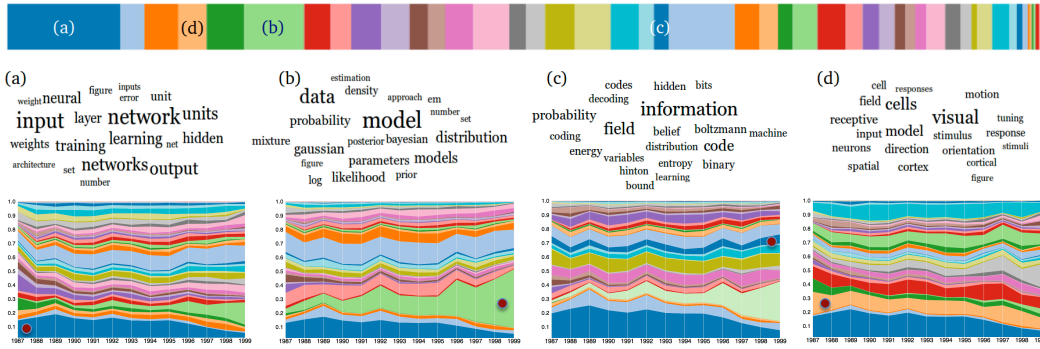

Figure 3: The DCNT predicts topic frequencies over the years (1987-1999) for documents with (a) none of the most prolific authors, (b) the Michael Jordan feature, (c) the Geoffrey Hinton feature, and (d) the Terrence Sejnowski feature. The stick-breaking distribution at the top shows the frequencies of each topic, averaging over all years; note some are unused. The middle row illustrates the word distributions for the topics highlighted by red dots in their respective columns. Larger words are more probable.

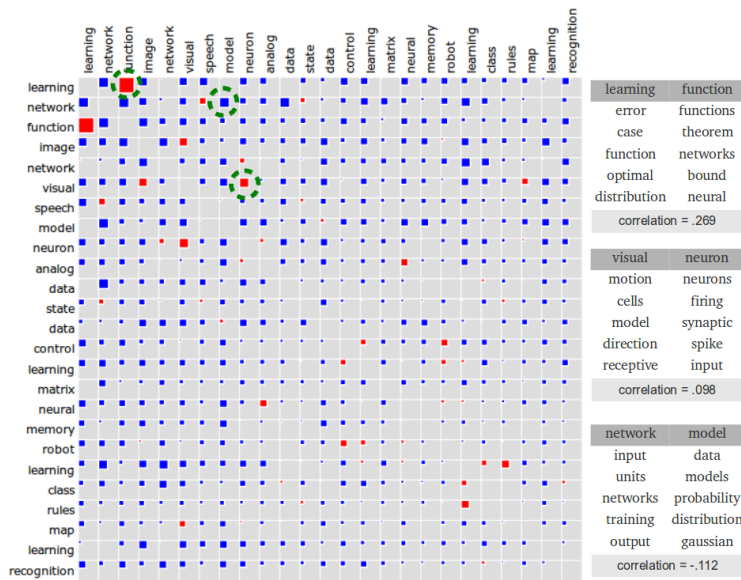

Figure 4: A Hinton diagram of correlations between all pairs of topics, where the sizes of squares indicates the magnitude of dependence, and red and blue squares indicate positive and negative correlations, respectively. To the right are the top six words from three strongly correlated topic pairs. This visualization, along with others in this paper, are interactive and can be downloaded from this page: `http://www.cs.brown.edu/~daeil`.

coefficients between two topics. The results displayed in this figure are for a model trained without metadata. We can see that the model learned strong positive correlations between *function* and *learning* topics which have strong semantic similarities, but are not identical. Another positive correlation that the model discovered was between the topics *visual* and *neuron*; of course there are many papers at NIPS which study the brain's visual cortex. A strong negative correlation was found between the *network* and *model* topics, which might reflect an idealogical separation between papers studying neural networks and probabilistic models.

### 4.3 Predictive Likelihood

In order to quantitatively measure the generalization power of our DCNT model, we tested several variants on two versions of the toy bars dataset (correlated & uncorrelated). We also compared models on the NIPS corpus, to explore more realistic data where metadata is available. The test data for the toy dataset consisted of 500 documents generated by the same process as the training data,

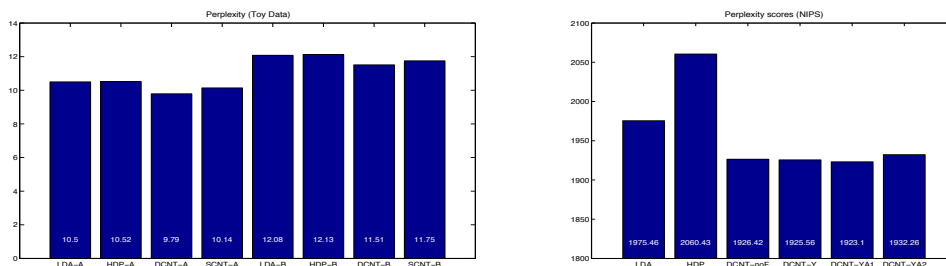

Figure 5: Perplexity scores (lower is better) computed via Chib-style estimators for several topic models. *Left:* Test performance for the toy datasets with uncorrelated bars (*-A*) and correlated bars (*-B*). *Right:* Test performance on the NIPS corpus with various metadata: no features (*-noF*), year features (*-Y*), year and prolific author features (over 10 publications, *-YA1*), and year and additional author features (over 5 publications, *-YA2*).

while the NIPS corpus was split into training and tests subsets containing 80% and 20% of the full corpus, respectively. Over the years 1988-1999, there were a total of 328 test documents.

We calculated predictive likelihood estimates using a Chib-style estimator [12]; for details see the supplemental material. In a previous comparison [19], the Chib-style estimator was found to be far more accurate than alternatives like the harmonic mean estimator. Note that there is some subtlety in correctly implementing the Chib-style estimator for our DCNT model, due to the possibility of rejection of our Metropolis-Hastings proposals.

Predictive negative log-likelihood estimates were normalized by word counts to determine perplexity scores [3]. We tested several models, including the SCNT and DCNT, LDA with $\alpha = 1$ and $\beta = 0.01$, and the HDP with full resampling of its concentration parameters. For the toy bars data, we set the number of topics to $K = 10$ for all models except the HDP, which learned $K = 15$. For the NIPS corpus, we set $K = 50$ for all models except the HDP, which learned $K = 86$.

For the toy datasets, the LDA and HDP models perform similarly. The SCNT and DCNT are both superior, apparently due to their ability to capture non-Dirichlet distributions on topic occurrence patterns. For the NIPS data, all of the DCNT models are substantially more accurate than LDA and the HDP. Including metadata encoding the year of publication, and possibly also the most prolific authors, provides slight additional improvements in DCNT accuracy. Interestingly, when a larger set of author features is included, accuracy becomes slightly worse. This appears to be an overfitting issue: there are 125 authors with over 5 publications, and only a handful of training examples for each one.

While it is pleasing that the DCNT and SCNT models seem to provide improved predictive likelihoods, a recent study on the human interpretability of topic models showed that such scores do not necessarily correlate with more meaningful semantic structures [4]. In many ways, the interactive visualizations illustrated in Sec. 4.2 provide more assurance that the DCNT can capture useful properties of real corpora.

# 5    Discussion

The doubly correlated nonparametric topic model flexibly allows the incorporation of arbitrary features associated with documents, captures correlations that might exist within a dataset's latent topics, and can learn an unbounded set of topics. The model uses a set of efficient MCMC techniques for learning and inference, and is supported by a set of web-based tools that allow users to visualize the inferred semantic structure.

# Acknowledgments

This research supported in part by IARPA under AFRL contract number FA8650-10-C-7059. Dae Il Kim supported in part by an NSF Graduate Fellowship. The views and conclusions contained herein are those of the authors and should not be interpreted as necessarily representing the official policies or endorsements, either expressed or implied, of IARPA, AFRL, or the U.S. Government.

## Footnotes

[1]One can exactly sample from a Dirichlet distribution by drawing a vector of independent gamma random variables, and normalizing so they sum to one. This normalization induces slight negative correlations.

[2] For any matrix $\eta$, we let $\eta_{:k}$ denote a column vector indexed by $k$, and $\eta_{f:}$ a row vector indexed by f.

# References

[1] A. Agovic and A. Banerjee. Gaussian process topic models. In *UAI*, 2010.

[2] D. M. Blei and J. D. Lafferty. A correlated topic model of science. *AAS*, 1(1):17–35, 2007.

[3] D. M. Blei, A. Y. Ng, and M. I. Jordan. Latent Dirichlet allocation. *J. Mach. Learn. Res.*, 3:993–1022, March 2003.

[4] J. Chang, J. Boyd-Graber, S. Gerrish, C. Wang, and D. M. Blei. Reading tea leaves: How humans interpret topic models. In *NIPS*, 2009.

[5] T. S. Ferguson. A Bayesian analysis of some nonparametric problems. *An. Stat.*, 1(2):209–230, 1973.

[6] A. Gelman, J. B. Carlin, H. S. Stern, and D. B. Rubin. *Bayesian Data Analysis*. Chapman & Hall, 2004.

[7] T. L. Griffiths and M. Steyvers. Finding scientific topics. *PNAS*, 2004.

[8] H. Ishwaran and L. F. James. Gibbs sampling methods for stick-breaking priors. *Journal of the American Statistical Association*, 96(453):161–173, Mar. 2001.

[9] W. Li, D. Blei, and A. McCallum. Nonparametric Bayes Pachinko allocation. In *UAI*, 2008.

[10] H. F. Lopes and M. West. Bayesian model assessment in factor analysis. *Stat. Sinica*, 14:41–67, 2004.

[11] D. Mimno and A. McCallum. Topic models conditioned on arbitrary features with dirichlet-multinomial regression. In *UAI*, 2008.

[12] I. Murray and R. Salakhutdinov. Evaluating probabilities under high-dimensional latent variable models. In *NIPS 21*, pages 1137–1144. 2009.

[13] J. Paisley, C. Wang, and D. Blei. The discrete infinite logistic normal distribution for mixed-membership modeling. In *AISTATS*, 2011.

[14] L. Ren, L. Du, L. Carin, and D. B. Dunson. Logistic stick-breaking process. *JMLR*, 12, 2011.

[15] A. Rodriguez and D. B. Dunson. Nonparametric bayesian models through probit stick-breaking processes. *J. Bayesian Analysis*, 2011.

[16] J. Sethuraman. A constructive definition of Dirichlet priors. *Stat. Sin.*, 4:639–650, 1994.

[17] Y. W. Teh, M. I. Jordan, M. J. Beal, and D. M. Blei. Hierarchical Dirichlet processes. *Journal of the American Statistical Association*, 101(476):1566–1581, 2006.

[18] Y. W. Teh, M. Seeger, and M. I. Jordan. Semiparametric latent factor models. In *AIStats 10*, 2005.

[19] H. M. Wallach, I. Murray, R. Salakhutdinov, and D. Mimno. Evaluation methods for topic models. In *ICML*, 2009.

